# On The Circuit Complexity of Neural Networks

**V. P. Roychowdhury**
Information Systems Laboratory
Stanford University
Stanford, CA, 94305

**K. Y. Siu**
Information Systems Laboratory
Stanford University
Stanford, CA, 94305

**A. Orlitsky**
AT&T Bell Laboratories
600 Mountain Avenue
Murray Hill, NJ, 07974

**T. Kailath**
Information Systems Laboratory
Stanford University
Stanford, CA, 94305

## Abstract

We introduce a geometric approach for investigating the power of threshold circuits. Viewing $n$-variable boolean functions as vectors in $\mathcal{R}^{2^n}$, we invoke tools from linear algebra and linear programming to derive new results on the realizability of boolean functions using threshold gates.

Using this approach, one can obtain: (1) upper-bounds on the number of spurious memories in Hopfield networks, and on the number of functions implementable by a depth-$d$ threshold circuit; (2) a lower bound on the number of orthogonal input functions required to implement a threshold function; (3) a necessary condition for an arbitrary set of input functions to implement a threshold function; (4) a lower bound on the error introduced in approximating boolean functions using sparse polynomials; (5) a limit on the effectiveness of the only known lower-bound technique (based on computing correlations among boolean functions) for the depth of threshold circuits implementing boolean functions, and (6) a constructive proof that every boolean function $f$ of $n$ input variables is a threshold function of polynomially many input functions, none of which is significantly correlated with $f$. Some of these results lead to generalizations of key results concerning threshold circuit complexity, particularly those that are based on the so-called spectral or Harmonic analysis approach. Moreover, our geometric approach yields simple proofs, based on elementary results from linear algebra, for many of these earlier results.

## 1   Introduction

An $S$-input threshold gate is characterized by $S$ real weights $w_1, \ldots, w_S$. It takes $S$ inputs: $x_1, \ldots, x_S$, each either $+1$ or $-1$, and outputs $+1$ if the linear combination $\sum_{i=1}^{S} w_i x_i$ is positive and $-1$ if the linear combination is negative. Threshold gates were recently used to implement several functions of practical interest (including: Parity, Addition, Multiplication, Division, and Comparison) with fewer gates and reduced depth than conventional circuits using AND, OR, and NOT gates [12, 4, 11].

This success has led to a considerable amount of research on the power of threshold circuits [1, 10, 9, 11, 3, 13]. However, even simple questions remain unanswered. It is not known, for example, whether there is a function that can be computed by a depth-3 threshold circuit with polynomially many gates but cannot be computed by any depth-2 circuit with polynomially many threshold gates.

Geometric approaches have proven useful for analyzing threshold gates. An $S$-input threshold gate corresponds to a hyperplane in $\mathcal{R}^S$. This has been used for example to count the number of boolean functions computable by a single threshold gate [6], and also to determine functions that cannot be implemented by a single threshold gate. However, threshold circuits of depth two or more do not carry a simple geometric interpretation in $\mathcal{R}^S$. The inputs to gates in the second level are themselves threshold functions, hence the linear combination computed at the second level is a non-linear function of the inputs. Lacking a geometric view, researchers [5, 3] have used indirect approaches, applying harmonic-analysis techniques to analyze threshold gates. These techniques, apart from their complexity, restricted the input functions of the gates to be of very special types: input variables or parities of the input variables, thus not applying even to depth-two circuits.

In this paper, we describe a simple geometric relation between the output function of a threshold gate and its set of input functions. This applies to arbitrary sets of input functions. Using this relation, we can prove the following results: (1) upper bounds on (a) the number of threshold functions of any set of input functions, (b) the number of spurious memories in a Hopfield network, and (c) the number of functions implementable by threshold circuits of depth $d$; (2) a lower bound on the number of orthogonal input functions required to implement a threshold function; (3) a quantifiable necessary condition for a set of functions to implement a threshold function; (4) a lower bound on the error in approximating boolean functions using sparse polynomials; (5) a limit on the effectiveness of the correlation method used in [7] to prove that a certain function cannot be implemented by depth two circuits with polynomially many gates and polynomially bounded weights; (6) a proof that every function $f$ is a threshold function of polynomially many input functions, none of which is significantly correlated with $f$.

Special cases of some of these results, where the input functions to a threshold gate are restricted to the input variables, or parities of the input variables, were proven in [5, 3] using harmonic-analysis tools. Our technique shows that these tools are not needed, providing simpler proofs for more general results.

Due to space limitations, we cannot present the full details of our results. Instead, we shall introduce the basic definitions followed by a technical summary of the results; the emphasis will be on pointing out the motivation and relating our results

with those in the literature. The proofs and other technical details will appear in a complete journal paper.

## 2   Definitions and Background

An *n-variable boolean function* is a mapping $f : \{-1,1\}^n \to \{-1,1\}$. We view $f$ as a (column) vector in $\mathcal{R}^{2^n}$. Each of $f$'s $2^n$ components is either $-1$ or $+1$ and represents $f(x)$ for a distinct value assignment $x$ of the $n$ boolean variables. We view the $S$ weights of an $S$-input threshold gate as a *weight vector* $\mathbf{w} = (w_1, \cdots, w_S)^T$ in $\mathcal{R}^S$.

Let the functions $f_1, \ldots, f_S$ be the inputs of a threshold gate $\mathbf{w}$. The gate *computes* a function $f$ (or $f$ is the *output* of the gate) if the following vector equation holds:

$$f = sgn\left(\sum_{i=1}^{S} f_i w_i\right) \tag{1}$$

where

$$sgn(x) = \begin{cases} +1 & \text{if } x > 0, \\ -1 & \text{if } x < 0, \\ \text{undefined} & \text{if } x = 0. \end{cases}$$

Note that this definition requires that *all components of $\sum_{i=1}^{S} f_i w_i$ be nonzero.* It is convenient to write Equation (1) in a matrix form:

$$f = sgn(Y\mathbf{w})$$

where the *input matrix*

$$Y = [f_1 \cdots f_S]$$

is a $2^n$ by $S$ matrix whose columns are the input functions. The function $f$, is a *threshold function* of $f_1, \ldots, f_S$ if there exists a threshold gate (i.e., $\mathbf{w}$) with inputs $f_1, \ldots, f_S$ that computes $f$.

These definitions form the basis of our approach. Each function, being a $\pm 1$ vector in $\mathcal{R}^{2^n}$, determines an *orthant* in $\mathcal{R}^{2^n}$. A function $f$ is the output of a threshold gate whose input functions are $f_1, \ldots, f_S$ if and only if the linear combination $\sum_{i=1}^{S} f_i w_i$ defined by the gate lies inside the orthant determined by $f$.

**Definition 1** *The correlation of two n-variable boolean functions $f_1$ and $f_2$ is:*

$$C_{f_1 f_2} = (f_1^T f_2)/2^n;$$

*the two functions are* uncorrelated or orthogonal *if $C_{f_1 f_2} = 0$.*

Note that $C_{f_1 f_2} = 1 - 2 d_H(f_1, f_2)/2^n$, where $d_H(f_1, f_2)$ is the Hamming distance between $f_1$ and $f_2$; thus, the correlation can be interpreted as a measure of how 'close' the two functions are.

Fix the input functions $f_1, \ldots f_S$ to a threshold gate. The *correlation vector* of a function $f$, with the input functions is

$$C_{fY} = (Y^T f)/2^n = (C_{ff_1}\ C_{ff_2}\ \cdots\ C_{ffs})^T.$$

Next, we define $\hat{C}$ as the maximum in magnitude among the correlation coefficients, i.e., $\hat{C} = \{|C_{ff_i}| : 1 \le i \le S\}$.

## 3   Summary of Results

The correlation between two $n$-variable functions is a multiple of $2^{-(n-1)}$, bounded between $-1$ and $1$, hence can assume $2^n + 1$ values. The correlation vector $C_{fY} = (C_{ff_1}, \ldots, C_{ff_1})^T$ can therefore assume at most $(2^n + 1)^S$ different values. There are $2^{2^n}$ Boolean functions of $n$ Boolean variables, hence many share the same correlation vector. However, the next theorem says that a threshold function of $f_1, \ldots, f_S$ does not share its correlation vector with any other function.

**Uniqueness Theorem**   *Let $f$ be a threshold function of $f_1, \ldots, f_S$. Then, for all $g \neq f$,*

$$C_{gY} \neq C_{fY}$$

**Corollary 1**   *There are at most $(2^n + 1)^S$ threshold functions of any set of $S$ input functions.*

The special case of the Uniqueness Theorem where the functions $f_1, \ldots, f_S$ are the input variables had been proven in [5, 9]. The proof used harmonic-analysis tools such as Parseval's theorem. It relied on the mutual orthogonality of the input functions (namely, $C_{x_i,x_j} = 0$ for all $i \neq j$). Another special case where the input functions are parities of the input variables was proven in [3]. The same proof was used; see *e.g.*, pages 419-422 of [9]. Our proof shows that the harmonic-analysis tools and assumptions are not needed thereby (1) significantly simplifying the proof, and (2) showing that the functions $f_1, \ldots, f_S$ need not be orthogonal: the Uniqueness Theorem holds for all collections of functions. The more general result of the Uniqueness Theorem can be applied to obtain the following two new counting results.

**Corollary 2**   *The number of stable states in a Hopfield network with $n$ elements which is programmed by the outer product rule to store $s$ given vectors is $\leq 2^{s \log(n+1)}$.*

**Corollary 3**   *Let $F_n(S(n), d)$ be the number of $n$-variable boolean functions computed by depth-$d$ threshold circuits with fan-in bounded by $S(n)$ (we assume $S(n) \geq n$). Then, for all $d, n \geq 1$,*

$$F_n(S(n), d) \leq 2^{O(n^2 S(n)^{d-1})} .$$

It follows easily from our geometric framework that if $C_{fY} = 0$ then $f$ is not a threshold function of $f_1, \ldots, f_S$: every linear combination of $f_1, \ldots, f_S$ is orthogonal to $f$, hence cannot intersect the orthant determined by $f$.

Next, we consider the case where $C_{fY} \neq 0$. Define the *generalized spectrum* to be the $S$-dimensional vector:

$$\beta = (\beta_1, \ldots, \beta_S)^T = (Y^T Y)^{-1} Y^T f$$

(the reason for the definition and the name will be clarified soon).

**Spectral-bound Theorem** *If $f$ is a linear threshold function of $f_1, \ldots, f_S$, then*

$$\sum_{i=1}^{S} |\beta_i| \geq 1, \quad hence,$$

$$S \geq 1/\hat{\beta}, \quad where \quad \hat{\beta} = \max\{|\beta_i| : 1 \leq i \leq S\}$$

The Spectral-Bound theorem provides a way of *lower bounding* the number $S$ of input functions. Specifically, if $\beta_i$ is exponentially small (in $n$) for all $i \in \{1, \ldots, S\}$, then $S$ must be exponentially large.

In the special case where the input functions are parities of the input variables, all input functions are orthogonal; hence $Y^T Y = 2^n I_S$ and

$$\beta = \frac{1}{2^n} Y^T f = C_{fY} \ .$$

Note that every parity function $p$ is a basis function of the Hadamard transform, hence $C_{fp}$ is the *spectral coefficient* corresponding to $p$ in the transform (see [8, 2] for more details on spectral representation of boolean functions). Therefore, the generalized spectrum in this case is the real spectrum of $f$. In that case, the Spectral-Bound Theorem implies that $S \geq \frac{1}{\max\{|C_{ff_i}|:1\leq i\leq S\}}$. Therefore, the number of input functions needed is at least the reciprocal of the maximum magnitude among the spectral coefficients (*i.e.*, $\hat{C}$). This special case was proved in [3]. Again, their proofs used harmonic-analysis tools and assumptions that we prove are unnecessary, thereby generalizing them to arbitrary input functions. Moreover, our geometric approach considerably simplifies the exposition by presenting simple proofs based on elementary results from linear algebra.

In general, we can show that if the input functions $f_i$ are orthogonal (*i.e.*, $C_{f_i f_j} = 0$ for $i \neq j$) or asymptotically orthogonal (*i.e.*, $\lim_{n \to \infty} C_{f_i f_j} = 0$) then the number of input functions $S \geq 1/\hat{C}$, where $\hat{C}$ is the largest (in magnitude) correlation of the output function with any of its input function.

We can also use the generalized spectrum to derive a lower bound on the *error* incurred in approximating a boolean function, $f$, using a set of basis functions. The lower bound can then be applied to show that the Majority function cannot be *closely approximated* by a sparse polynomial. In particular, it can be shown that if a polynomial of the input variables with only polynomially many (in $n$) monomials is used to approximate an $n$ variable Majority function then the approximation error is $\Omega(1/(\log\log n)^{3/2})$. This provides a direct spectral approach for proving lower bounds on the approximation error.

The method of proving lower bounds on $S$ in terms of the correlation coefficients $C_{ff_i}$ of $f$ with the possible input functions, can be termed the *method of correlations*. Hajnal et. al. [7] used a different aspect of this method[1] to prove a lower bound on the depth of a threshold circuit that computes the Inner-product-mod-2 function.

Our techniques can be applied to investigate the method of correlations in more detail and prove some limits to its effectiveness. We can show that the number, $S$, of input functions need not be inversely proportional to the largest correlation coefficient $\hat{C}$. In particular, we give two *constructive procedures* showing that any function $f$ is a threshold function of $O(n)$ input functions each having an exponentially small correlation with $f$: $|C_{ff_i}| \leq 2^{-(n-1)}$.

**Construction 1**   *Every boolean function $f$ of $n$ variables (for $n$ even) can be expressed as a threshold function of $3n$ boolean functions: $f_1, f_2, \cdots, f_{3n}$ such that (1) $C_{ff_i} = 0$, $\forall 1 \leq i \leq 3n - 1$, and (2) $C_{ff_{3n}} = 2^{-(n-1)}$.*

**Construction 2**   *Every boolean function $f$ of $n$ variables can be expressed as a threshold function of $2n$ boolean functions: $f_1, f_2, \cdots, f_{2n}$ such that (1) $C_{ff_i} = 0$, $\forall 1 \leq i \leq 2n - 2$, and (2) $C_{ff_{2n-1}} = C_{ff_{2n}} = 2^{-(n-1)}$.*

The results of the above constructions are surprising. For example, in Construction 1, the output function of the threshold gate is uncorrelated with all but one of the input functions, and the only non-zero correlation is the smallest possible ($= 2^{-(n-1)}$). Note that $f$ is not a threshold function of a set of input functions, each of which is orthogonal to $f$.

The above results thus provide a comprehensive understanding of the so-called method of correlations. In particular: (1) If the input functions are mutually orthogonal (or asymptotically orthogonal), then the method of correlations is effective even if exponential weights are allowed, *i.e.*, if a function is exponentially small correlated with every function from a pool of possible input functions, then one would require exponentially many inputs to implement the given function using a threshold gate; (2) If the input functions are not mutually orthogonal, then the method of correlations need not be effective, *i.e.*, one can construct examples, where the output function is correlated exponentially small with every input function, and yet it can be implemented as a threshold function of polynomially many input functions.

Furthermore, the constructive procedures can also be considered as constituting a *preliminary* answer to the following question: Given an $n$-variable boolean function $f$, are there efficient procedures for expressing it as threshold functions of polynomially many (in $n$) input functions? A procedure for so decomposing a given function $f$ will be referred to as a *threshold-decomposition* procedure; moreover, a decomposition procedure can be considered as efficient if the input functions have simpler threshold implementations than $f$ (i.e., easier to implement or require less depth/size). Constructions 1 and 2 present two such threshold-decomposition procedures. At present, the efficiency of these constructions is not clear and further work is necessary. We hope, however, that the general methodology introduced here may lead to subsequent work resulting in more efficient threshold-decomposition procedures.

# 4   Concluding Remarks

We have outlined a new geometric approach for investigating the properties of threshold circuits. In the process, we have developed a unified framework where many of the previous results can be derived simply as special cases, and without in-

troducing too many seemingly difficult concepts. Moreover, we have derived several new results that quantify the input/output relationships of threshold gates, derive lower bounds on the number of input functions required to implement a given function using a threshold gate, and also analyze the limitations of a well-known lower bound technique for threshold circuit.

## Acknowledgements

This work was supported in part by the Joint Services Program at Stanford University (US Army, US Navy, US Air Force) under Contract DAAL03-88-C-0011, the SDIO/IST, managed by the Army Research Office under Contract DAAL03-90-G-0108, and the Department of the Navy, NASA Headquarters, Center for Aeronautics and Space Information Sciences under Grant NAGW-419-S6.

## Footnotes

[1] They did not exactly use the correlation approach introduced in this paper, rather an equivalent framework.

## References

[1] E. Allender. A note on the power of threshold circuits. *IEEE Symp. Found. Comp. Sci.*, 30, 1989.

[2] Y. Bradman, A. Orlitsky, and J. Hennessy. A Spectral Lower Bound Technique for the size of Decision Trees and Two level AND/OR Circuits. *IEEE Trans. on Computers*, 39, No. 2:282–287, February 1990.

[3] J. Bruck. Harmonic Analysis of Polynomial Threshold Functions. *SIAM Journal on Discrete Mathematics*, May 1990.

[4] A. K. Chandra, L. Stockmeyer, and U. Vishkin. Constant depth reducibility. *Siam J. Comput.*, 13:423–439, 1984.

[5] C. K. Chow. On The Characterization of Threshold Functions. *Proc. Symp. on Switching Circuit Theory and Logical Design*, pages 34–38, 1961.

[6] T. M. Cover. Geometrical and Statistical Properties of Systems of Linear Inequalities with Applications in Pattern Recognition. *IEEE Trans. on Electronic Computers*, EC-14:326–34, 1965.

[7] A. Hajnal, W. Maass, P. Pudlak, M. Szegedy, and G. Turan. Threshold circuits of bounded depth. *IEEE Symp. Found. Comp. Sci.*, 28:99–110, 1987.

[8] R. J. Lechner. *Harmonic analysis of switching functions*. In A. Mukhopadhyay, editor, Recent Development in Switching Theory. Academic Press, 1971.

[9] P. M. Lewis and C. L. Coates. *Threshold Logic*. John Wiley & Sons, Inc., 1967.

[10] I. Parberry and G. Schnitger. Parallel Computation with Threshold Functions. *Journal of Computer and System Sciences*, 36(3):278–302, 1988.

[11] J. Reif. On Threshold Circuits and Polynomial Computation. In *Structure in Complexity Theory Symp.*, pages 118–123, 1987.

[12] K. Y. Siu and J. Bruck. On the Power of Threshold Circuits with Small Weights. to appear in SIAM J. Discrete Math.

[13] K. Y. Siu, V. P. Roychowdhury, and T. Kailath. Computing with Almost Optimal Size Threshold Circuits. submitted to JCSS, 1990.


